# Temporal Difference Learning in Continuous Time and Space

**Kenji Doya**
doya@hip.atr.co.jp
ATR Human Information Processing Research Laboratories
2-2 Hikaridai, Seika-cho, Soraku-gun, Kyoto 619-02, Japan

## Abstract

A continuous-time, continuous-state version of the temporal difference (TD) algorithm is derived in order to facilitate the application of reinforcement learning to real-world control tasks and neurobiological modeling. An optimal nonlinear feedback control law was also derived using the derivatives of the value function. The performance of the algorithms was tested in a task of swinging up a pendulum with limited torque. Both the "critic" that specifies the paths to the upright position and the "actor" that works as a nonlinear feedback controller were successfully implemented by radial basis function (RBF) networks.

## 1 INTRODUCTION

The temporal-difference (TD) algorithm (Sutton, 1988) for delayed reinforcement learning has been applied to a variety of tasks, such as robot navigation, board games, and biological modeling (Houk et al., 1994). Elucidation of the relationship between TD learning and dynamic programming (DP) has provided good theoretical insights (Barto et al., 1995). However, conventional TD algorithms were based on discrete-time, discrete-state formulations. In applying these algorithms to control problems, time, space and action had to be appropriately discretized using a priori knowledge or by trial and error. Furthermore, when a TD algorithm is used for neurobiological modeling, discrete-time operation is often very unnatural.

There have been several attempts to extend TD-like algorithms to continuous cases. Bradtke et al. (1994) showed convergence results for DP-based algorithms for a discrete-time, continuous-state linear system with a quadratic cost. Bradtke and Duff (1995) derived TD-like algorithms for continuous-time, discrete-state systems (semi-Markov decision problems). Baird (1993) proposed the "advantage updating" algorithm by modifying Q-learning so that it works with arbitrary small time steps.

In this paper, we derive a TD learning algorithm for continuous-time, continuous-state, nonlinear control problems. The correspondence of the continuous-time version to the conventional discrete-time version is also shown. The performance of the algorithm was tested in a nonlinear control task of swinging up a pendulum with limited torque.

## 2    CONTINUOUS-TIME TD LEARNING

We consider a continuous-time dynamical system (plant)

$$\dot{\mathbf{x}}(t) = f(\mathbf{x}(t), \mathbf{u}(t)) \tag{1}$$

where $\mathbf{x} \in X \subset \mathbf{R}^n$ is the state and $\mathbf{u} \in U \subset \mathbf{R}^m$ is the control input (action). We denote the immediate reinforcement (evaluation) for the state and the action as

$$r(t) = r(\mathbf{x}(t), \mathbf{u}(t)). \tag{2}$$

Our goal is to find a feedback control law (policy)

$$\mathbf{u}(t) = \mu(\mathbf{x}(t)) \tag{3}$$

that maximizes the expected reinforcement for a certain period in the future. To be specific, for a given control law $\mu$, we define the "value" of the state $\mathbf{x}(t)$ as

$$V^\mu(\mathbf{x}(t)) = \int_t^\infty \frac{1}{\tau} e^{-\frac{s-t}{\tau}} r(\mathbf{x}(s), \mathbf{u}(s)) ds, \tag{4}$$

where $\mathbf{x}(s)$ and $\mathbf{u}(s)$ $(t < s < \infty)$ follow the system dynamics (1) and the control law (3). Our problem now is to find an optimal control law $\mu^*$ that maximizes $V^\mu(\mathbf{x})$ for any state $\mathbf{x} \in X$. Note that $\tau$ is the time scale of "imminence-weighting" and the scaling factor $\frac{1}{\tau}$ is used for normalization, i.e., $\int_t^\infty \frac{1}{\tau} e^{-\frac{s-t}{\tau}} ds = 1$.

### 2.1    TD ERROR

The basic idea in TD learning is to predict future reinforcement in an on-line manner. We first derive a local consistency condition for the value function $V^\mu(\mathbf{x})$. By differentiating (4) by $t$, we have

$$\tau \frac{d}{dt} V^\mu(\mathbf{x}(t)) = V^\mu(\mathbf{x}(t)) - r(t). \tag{5}$$

Let $P(t)$ be the prediction of the value function $V^\mu(\mathbf{x}(t))$ from $\mathbf{x}(t)$ (output of the "critic"). If the prediction is perfect, it should satisfy $\tau \dot{P}(t) = P(t) - r(t)$. If this is not satisfied, the prediction should be adjusted to decrease the inconsistency

$$\hat{r}(t) = r(t) - P(t) + \tau \dot{P}(t). \tag{6}$$

This is a continuous version of the temporal difference error.

### 2.2    EULER DIFFERENTIATION: TD(0)

The relationship between the above continuous-time TD error and the discrete-time TD error (Sutton, 1988)

$$\hat{r}(t) = r(t) + \gamma P(t) - P(t - \Delta t) \tag{7}$$

can be easily seen by a backward Euler approximation of $\dot{P}(t)$. By substituting $\dot{P}(t) = (P(t) - P(t - \Delta t))/\Delta t$ into (6), we have

$$\hat{r} = r(t) + \frac{\tau}{\Delta t} \left[ (1 - \frac{\Delta t}{\tau}) P(t) - P(t - \Delta t) \right].$$

This coincides with (7) if we make the "discount factor" $\gamma = 1 - \frac{\Delta t}{\tau} \simeq e^{-\frac{\Delta t}{\tau}}$, except for the scaling factor $\frac{\tau}{\Delta t}$.

Now let us consider a case when the prediction of the value function is given by

$$P(t) = \sum_i v_i b_i(\mathbf{x}(t)), \tag{8}$$

where $b_i()$ are basis functions (e.g., sigmoid, Gaussian, etc) and $v_i$ are the weights. The gradient descent of the squared TD error is given by

$$\Delta v_i \propto -\frac{\partial \frac{1}{2}\hat{r}^2(t)}{\partial v_i} \propto -\hat{r}(t)\left[(1 - \frac{\Delta t}{\tau})\frac{\partial P(t)}{\partial v_i} - \frac{\partial P(t - \Delta t)}{\partial v_i}\right].$$

In order to "back-up" the information about the future reinforcement to correct the prediction in the past, we should modify $P(t - \Delta t)$ rather than $P(t)$ in the above formula. This results in the learning rule

$$\Delta v_i \propto \hat{r}(t)\frac{\partial P(t - \Delta t)}{\partial v_i} = \hat{r}(t)b_i(\mathbf{x}(t - \Delta t)). \tag{9}$$

This is equivalent to the TD(0) algorithm that uses the "eligibility trace" from the previous time step.

## 2.3 SMOOTH DIFFERENTIATION: TD($\lambda$)

The Euler approximation of a time derivative is susceptible to noise (e.g., when we use stochastic control for exploration). Alternatively, we can use a "smooth" differentiation algorithm that uses a weighted average of the past input, such as

$$\dot{P}(t) \simeq \frac{P(t) - \bar{P}(t)}{\tau_c} \qquad \text{where} \qquad \tau_c \frac{d}{dt}\bar{P}(t) = P(t) - \bar{P}(t)$$

and $\tau_c$ is the time constant of the differentiation. The corresponding gradient descent algorithm is

$$\Delta v_i \propto -\frac{\partial \frac{1}{2}\hat{r}^2(t)}{\partial v_i} \propto \hat{r}(t)\frac{\partial \bar{P}(t)}{\partial v_i} = \hat{r}(t)\bar{b}_i(t), \tag{10}$$

where $\bar{b}_i$ is the eligibility trace for the weight

$$\tau_c \frac{d}{dt}\bar{b}_i(t) = b_i(\mathbf{x}(t)) - \bar{b}_i(t). \tag{11}$$

Note that this is equivalent to the TD($\lambda$) algorithm (Sutton, 1988) with $\lambda = 1 - \frac{\Delta t}{\tau_c}$ if we discretize the above equation with time step $\Delta t$.

# 3 OPTIMAL CONTROL BY VALUE GRADIENT

## 3.1 HJB EQUATION

The value function $V^*$ for an optimal control $\mu^*$ is defined as

$$V^*(\mathbf{x}(t)) = \max_{\mathbf{u}[t,\infty)}\left[\int_t^\infty \frac{1}{\tau}e^{-\frac{s-t}{\tau}}r(\mathbf{x}(s), \mathbf{u}(s))ds\right]. \tag{12}$$

According to the principle of dynamic programming (Bryson and Ho, 1975), we consider optimization in two phases, $[t, t + \Delta t]$ and $[t + \Delta t, \infty)$, resulting in the expression

$$V^*(\mathbf{x}(t)) = \max_{\mathbf{u}[t,t+\Delta t)}\left[\int_t^{t+\Delta t} \frac{1}{\tau}e^{-\frac{s-t}{\tau}}r(\mathbf{x}(s), \mathbf{u}(s))ds + e^{-\frac{\Delta t}{\tau}}V^*(\mathbf{x}(t + \Delta t))\right].$$

By Taylor expanding the value at $t + \Delta t$ as

$$V^*(\mathbf{x}(t + \Delta t)) = V^*(\mathbf{x}(t)) + \frac{\partial V^*}{\partial \mathbf{x}(t)} f(\mathbf{x}(t), \mathbf{u}(t)) \Delta t + O(\Delta t)$$

and then taking $\Delta t$ to zero, we have a differential constraint for the optimal value function

$$V^*(t) = \max_{\mathbf{u}(t) \in U} \left[ r(\mathbf{x}(t), \mathbf{u}(t)) + \tau \frac{\partial V^*}{\partial \mathbf{x}} f(\mathbf{x}(t), \mathbf{u}(t)) \right]. \tag{13}$$

This is a variant of the Hamilton-Jacobi-Bellman equation (Bryson and Ho, 1975) for a discounted case.

## 3.2   OPTIMAL NONLINEAR FEEDBACK CONTROL

When the reinforcement $r(\mathbf{x}, \mathbf{u})$ is convex with respect to the control $\mathbf{u}$, and the vector field $f(\mathbf{x}, \mathbf{u})$ is linear with respect to $\mathbf{u}$, the optimization problem in (13) has a unique solution. The condition for the optimal control is

$$\frac{\partial r(\mathbf{x}, \mathbf{u})}{\partial \mathbf{u}} + \tau \frac{\partial V^*}{\partial \mathbf{x}} \frac{\partial f(\mathbf{x}, \mathbf{u})}{\partial \mathbf{u}} = 0. \tag{14}$$

Now we consider the case when the cost for control is given by a convex potential function $G_j()$ for each control input

$$f(\mathbf{x}, \mathbf{u}) = r_x(\mathbf{x}) - \sum_j G_j(u_j),$$

where reinforcement for the state $r_x(\mathbf{x})$ is still unknown. We also assume that the input gain of the system

$$\mathbf{b}_j(\mathbf{x}) = \frac{\partial f(\mathbf{x}, \mathbf{u})}{\partial u_j}$$

is available. In this case, the optimal condition (14) for $u_j$ is given by

$$-G_j'(u_j) + \tau \frac{\partial V^*}{\partial \mathbf{x}} \mathbf{b}_j(\mathbf{x}) = 0.$$

Noting that the derivative $G'()$ is a monotonic function since $G()$ is convex, we have the optimal feedback control law

$$u_j = (G')^{-1} \left( \tau \frac{\partial V^*}{\partial \mathbf{x}} \mathbf{b}(\mathbf{x}) \right). \tag{15}$$

Particularly, when the amplitude of control is bounded as $|u_j| \leq u_j^{\max}$, we can enforce this constraint using a control cost

$$G_j(u_j) = c_j \int_0^{\frac{u_j}{u_j^{\max}}} g^{-1}(s) ds, \tag{16}$$

where $g^{-1}()$ is an inverse sigmoid function that diverges at $\pm 1$ (Hopfield, 1984). In this case, the optimal feedback control law is given by

$$u_j = u_j^{\max} g \left( \frac{u_j^{\max}}{c_j} \tau \frac{\partial V^*}{\partial \mathbf{x}} \mathbf{b}_j(\mathbf{x}) \right). \tag{17}$$

In the limit of $c_j \to 0$, this results in the "bang-bang" control law

$$u_j = u_j^{\max} \operatorname{sign} \left[ \frac{\partial V^*}{\partial \mathbf{x}} \mathbf{b}_j(\mathbf{x}) \right]. \tag{18}$$

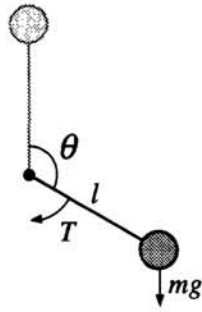

Figure 1: A pendulum with limited torque. The dynamics is given by $ml\ddot{\theta} = -\mu\dot{\theta} + mgl\sin\theta + T$. Parameters were $m = l = 1$, $g = 9.8$, and $\mu = 0.01$.

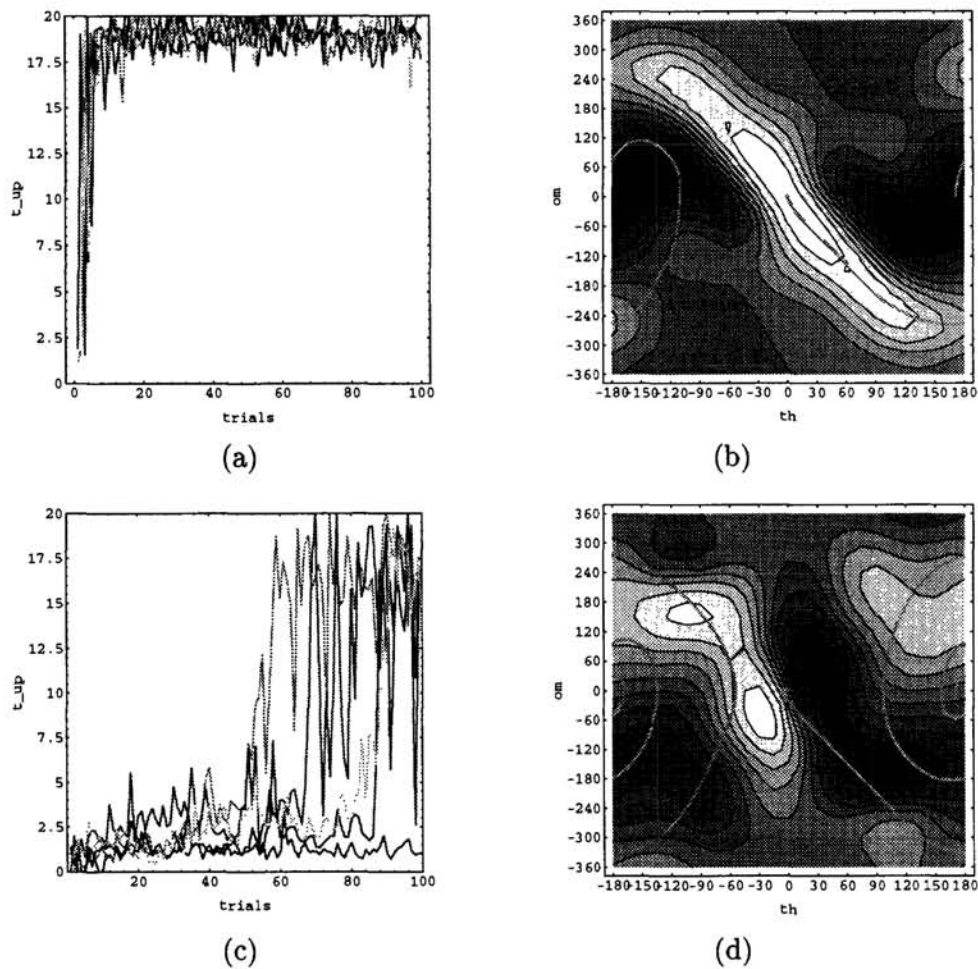

Figure 2: Left: The learning curves for (a) optimal control and (c) actor-critic. t_up: time during which $|\theta| < 90°$. Right: (b) The predicted value function $P$ after 100 trials of optimal control. (d) The output of the controller after 100 trials with actor-critic learning. The thick gray line shows the trajectory of the pendulum. th: $\theta$ (degrees), om: $\dot{\theta}$ (degrees/sec).

# 4 ACTOR-CRITIC

When the information about the control cost, the input gain of the system, or the gradient of the value function is not available, we cannot use the above optimal control law. However, the TD error (6) can be used as "internal reinforcement" for training a stochastic controller, or an "actor" (Barto et al., 1983).

In the simulation below, we combined our TD algorithm for the critic with a reinforcement learning algorithm for real-valued output (Gullapalli, 1990). The output of the controller was given by

$$u_j(t) = u_j^{\max} g \left( \sum_i w_{ji} b_i(\mathbf{x}(t)) + \sigma n_j(t) \right), \qquad (19)$$

where $n_j(t)$ is normalized Gaussian noise and $w_{ji}$ is a weight. The size of this perturbation was changed based on the predicted performance by $\sigma = \sigma_0 \exp(-P(t))$. The connection weights were changed by

$$\Delta w_{ji} \propto \hat{r}(t) n_j(t) b_i(\mathbf{x}(t)). \qquad (20)$$

# 5 SIMULATION

The performance of the above continuous-time TD algorithm was tested on a task of swinging up a pendulum with limited torque (Figure 1). Control of this one-degree-of-freedom system is trivial near the upright equilibrium. However, bringing the pendulum near the upright position is not if we set the maximal torque $T^{\max}$ smaller than $mgl$. The controller has to swing the pendulum several times to build up enough momentum to bring it upright. Furthermore, the controller has to decelerate the pendulum early enough to avoid falling over.

We used a radial basis function (RBF) network to approximate the value function for the state of the pendulum $\mathbf{x} = (\theta, \dot{\theta})$. We prepared a fixed set of $12 \times 12$ Gaussian basis functions. This is a natural extension of the "boxes" approach previously used to control inverted pendulums (Barto et al., 1983). The immediate reinforcement was given by the height of the tip of the pendulum, i.e., $r_x = \cos \theta$.

## 5.1 OPTIMAL CONTROL

First, we used the optimal control law (17) with the predicted value function $P$ instead of $V^*$. We added noise to the control command to enhance exploration. The torque was given by

$$T = T^{\max} g \left( \frac{T^{\max}}{c} \tau \frac{\partial P(\mathbf{x})}{\partial \mathbf{x}} \mathbf{b} + \sigma n(t) \right),$$

where $g(x) = \frac{2}{\pi} \tan^{-1}(\frac{\pi}{2} x)$ (Hopfield, 1984). Note that the input gain $\mathbf{b} = (0, 1/ml^2)^T$ was constant. Parameters were $T^{\max} = 5$, $c = 0.1$, $\sigma_0 = 0.01$, $\tau = 1.0$, and $\tau_c = 0.1$.

Each run was started from a random $\theta$ and was continued for 20 seconds. Within ten trials, the value function $P$ became accurate enough to be able to swing up and hold the pendulum (Figure 2a). An example of the predicted value function $P$ after 100 trials is shown in Figure 2b. The paths toward the upright position, which were implicitly determined by the dynamical properties of the system, can be seen as the ridges of the value function. We also had successful results when the reinforcement was given only near the goal: $r_x = 1$ if $|\theta| < 30°$, $-1$ otherwise.

## 5.2  ACTOR-CRITIC

Next, we tested the actor-critic learning scheme as described above. The controller was also implemented by a RBF network with the same $12 \times 12$ basis functions as the critic network. It took about one hundred trials to achieve reliable performance (Figure 2c). Figure 2d shows an example of the output of the controller after 100 trials. We can see nearly linear feedback in the neighborhood of the upright position and a non-linear torque field away from the equilibrium.

## 6  CONCLUSION

We derived a continuous-time, continuous-state version of the TD algorithm and showed its applicability to a nonlinear control task. One advantage of continuous formulation is that we can derive an explicit form of optimal control law as in (17) using derivative information, whereas a one-ply search for the best action is usually required in discrete formulations.

### References

Baird III, L. C. (1993). Advantage updating. Technical Report WL-TR-93-1146, Wright Laboratory, Wright-Patterson Air Force Base, OH 45433-7301, USA.

Barto, A. G., Bradtke, S. J., and Singh, S. P. (1995). Learning to act using real-time dynamic programming. *Artificial Intelligence*, 72:81–138.

Barto, A. G., Sutton, R. S., and Anderson, C. W. (1983). Neuronlike adaptive elements that can solve difficult learning control problems. *IEEE Transactions on System, Man, and Cybernetics*, SMC-13:834–846.

Bradtke, S. J. and Duff, M. O. (1995). Reinforcement learning methods for continuous-time Markov decision problems. In Tesauro, G., Touretzky, D. S., and Leen, T. K., editors, *Advances in Neural Information Processing Systems 7*, pages 393–400. MIT Press, Cambridge, MA.

Bradtke, S. J., Ydstie, B. E., and Barto, A. G. (1994). Adaptive linear quadratic control using policy iteration. CMPSCI Technical Report 94-49, University of Massachusetts, Amherst, MA.

Bryson, Jr., A. E. . and Ho, Y.-C. (1975). *Applied Optimal Control*. Hemisphere Publishing, New York, 2nd edition.

Gullapalli, V. (1990). A stochastic reinforcement learning algorithm for learning real-valued functions. *Neural Networks*, 3:671–192.

Hopfield, J. J. (1984). Neurons with graded response have collective computational properties like those of two-state neurons. *Proceedings of National Academy of Science*, 81:3088–3092.

Houk, J. C., Adams, J. L., and Barto, A. G. (1994). A model of how the basal ganglia generate and use neural signlas that predict renforcement. In Houk, J. C., Davis, J. L., and Beiser, D. G., editors, *Models of Information Processing in the Basal Ganglia*, pages 249–270. MIT Press, Cambrigde, MA.

Sutton, R. S. (1988). Learning to predict by the methods of temporal difference. *Machine Learning*, 3:9–44.
